# Optimal signalling in Attractor Neural Networks

**Isaac Meilijson**          **Eytan Ruppin** *
School of Mathematical Sciences
Raymond and Beverly Sackler Faculty of Exact Sciences
Tel-Aviv University, 69978 Tel-Aviv, Israel.

## Abstract

In [Meilijson and Ruppin, 1993] we presented a methodological framework describing the two-iteration performance of Hopfield-like attractor neural networks with history-dependent, Bayesian dynamics. We now extend this analysis in a number of directions: input patterns applied to small subsets of neurons, general connectivity architectures and more efficient use of history. We show that the optimal signal (activation) function has a *slanted sigmoidal* shape, and provide an intuitive account of activation functions with a non-monotone shape. This function endows the model with some properties characteristic of cortical neurons' firing.

## 1  Introduction

It is well known that a given cortical neuron can respond with a different firing pattern for the same synaptic input, depending on its firing history and on the effects of modulator transmitters (see [Connors and Gutnick, 1990] for a review). The time span of different channel conductances is very broad, and the influence of some ionic currents varies with the history of the membrane potential [Lytton, 1991]. Motivated by the history-dependent nature of neuronal firing, we continue.our previous investigation [Meilijson and Ruppin, 1993] (henceforth, M & R) describing the performance of Hopfield-like attractor neural networks (ANN) [Hopfield, 1982] with history-dependent dynamics.

Building upon the findings presented in M & R, we now study a more general framework:

- We differentiate between 'input' neurons receiving the initial input signal with high fidelity and 'background' neurons that receive it with low fidelity.

- Dynamics now depend on the neuron's history of firing, in addition to its history of input fields.

- The dependence of ANN performance on the network architecture can be explicitly expressed. In particular, this enables the investigation of cortical-like architectures, where neurons are randomly connected to other neurons, with higher probability of connections formed between spatially proximal neurons [Braitenberg and Schuz, 1991].

Our goal is twofold: first, to search for the computationally most efficient history-dependent neuronal signal (firing) function, and study its performance with relation to memoryless dynamics. As we shall show, optimal history-dependent dynamics are indeed much more efficient than memoryless ones. Second, to examine the optimal signal function from a biological perspective. As will shall see, it shares some basic properties with the firing of cortical neurons.

## 2    The model

Our framework is an ANN storing $m + 1$ memory patterns $\xi^1, \xi^2, \ldots, \xi^{m+1}$, each an N-dimensional vector. The network is composed of $N$ neurons, each of which is randomly connected to $K$ other neurons. The $(m + 1)N$ memory entries are independent with equally likely $\pm 1$ values. The initial pattern $X$, signalled by $L(\leq N)$ *initially active* neurons, is a vector of $\pm 1$'s, randomly generated from one of the memory patterns (say $\xi = \xi^{m+1}$) such that $P(X_i = \xi_i) = \frac{1 \pm \epsilon}{2}$ for each of the $L$ initially active neurons and $P(X_i = \xi_i) = \frac{1 \pm \delta}{2}$ for each *initially quiescent* (non-active) neuron. Although $\epsilon, \delta \in [0, 1)$ are arbitrary, it is useful to think of $\epsilon$ as being 0.5 (corresponding to an initial similarity of 75%) and of $\delta$ as being zero - a quiescent neuron has no prior preference for any given sign. Let $\alpha_1 = m/n_1$ denote the initial memory load, where $n_1 = LK/N$ is the average number of signals received by each neuron.

The notion of 'iteration' is viewed as an abstraction of the overall dynamics for some length of time, during which some continuous input/output signal function (such as the conventional sigmoidal function) governs the firing rate of the neuron. We follow a Bayesian approach under which the neuron's signalling and activation decisions are based on the a-posteriori probabilities assigned to its two possible true memory states, $\pm 1$.

Initially, neuron $i$ is assigned a prior probability $\lambda_i^{(0)} = P(\xi = 1 | X_i, I_i^{(1)}) = \frac{1 \pm \epsilon}{2}$ or $\frac{1 \pm \delta}{2}$ which is conveniently expressed as $\lambda_i^{(0)} = \frac{1}{1 + e^{-2g_i(0)}}$, where, letting $g(t) = \frac{1}{2} \log \frac{1+t}{1-t}$,

$$g_i^{(0)} = \begin{cases} g(\epsilon)X_i & \text{if } i \text{ is active} \\ g(\delta)X_i & \text{if } i \text{ is silent} \end{cases}$$

The input field observed by neuron $i$ as a result of the initial activity is

$$f_i^{(1)} = \frac{1}{n_1} \sum_{j=1}^{N} W_{ij} I_{ij} I_j^{(1)} X_j \tag{1}$$

where $I_j^{(1)} = 0, 1$ indicates whether neuron $j$ has fired in the first iteration, $I_{ij} = 0, 1$ indicates whether a connection exists from neuron $j$ to neuron $i$ and $W_{ij}$ denotes its magnitude, given by the Hopfield prescription

$$W_{ij} = \sum_{\mu=1}^{m+1} \xi^\mu{}_i \xi^\mu{}_j \,. \tag{2}$$

As a result of observing the input field $f_i^{(1)}$, which is approximately normally distributed (given $\xi_i, X_i$ and $I_i^{(1)}$), neuron $i$ changes its opinion about $\{\xi_i = 1\}$ from $\lambda_i^{(0)}$ to

$$\lambda_i^{(1)} = P\left(\xi_i = 1 | X_i, I_i^{(1)}, f_i^{(1)}\right) = \frac{1}{1 + e^{-2g_i^{(1)}}} \,, \tag{3}$$

expressed in terms of the (additive) generalized field $g_i^{(1)} = g_i^{(0)} + \frac{\epsilon}{\alpha_1} f_i^{(1)}$.

We now get to the second iteration, in which, as in the first iteration, some of the neurons become active and signal to the network. We model the signal function neuron $i$ emits as $h(g_i^{(1)}, X_i, I_i^{(1)})$. The field observed by neuron $i$ (with $n_2$ updating neurons per neuron) is

$$f_i^{(2)} = \frac{1}{n_2} \sum_{j=1}^{N} W_{ij} I_{ij} h(g_j^{(1)}, X_j, I_j^{(1)}) \,, \tag{4}$$

on the basis of which neuron $i$ computes its posterior belief $\lambda_i^{(2)} = P(\xi_i = 1 | X_i, I_i^{(1)}, f_i^{(1)}, f_i^{(2)})$ and expresses its final choice of sign as $X_i^{(2)} = sign(\lambda_i^{(2)} - 0.5)$. The two-iteration performance of the network is measured by the final similarity

$$S_f = \frac{1 + \epsilon_f}{2} = P(X_i^{(2)} = \xi_i) = \frac{1 + \frac{1}{N}\sum_{j=1}^{N} X_j^{(2)} \xi_j}{2} \,. \tag{5}$$

## 3   Analytical results

The goals of our analysis have been: A. To present an expression for the performance under arbitrary architecture and activity parameters, for general signal functions $h_0$ and $h_1$. B. Use this expression to find the best choice of signal functions which maximize performance. We show the following:

The neuron's final decision is given by

$$X_i^{(2)} = Sign\left[(A_0 + B_0 I_i^{(1)})X_i + A_1 f_i^{(1)} + A_2 f_i^{(2)}\right] \tag{6}$$

for some constants $A_0$, $B_0$, $A_1$ and $A_2$.

The performance achieved is

$$\frac{n_1}{K}(Q^*\Phi)\left(\frac{\epsilon}{\sqrt{\alpha^*}},\epsilon\right) + \left(1 - \frac{n_1}{K}\right)(Q^*\Phi)\left(\frac{\epsilon}{\sqrt{\alpha^*}},\delta\right) \qquad (7)$$

where, for some $A_3 > 0$

$$\alpha^* = \frac{m}{n^*} = \frac{m}{n_1 + mA_3}, \qquad (8)$$

$$(Q^*\Phi)(x,t) = \frac{1+t}{2}\Phi\left(x + \frac{g(t)}{x}\right) + \frac{1-t}{2}\Phi\left(x - \frac{g(t)}{x}\right) \qquad (9)$$

and $\Phi$ is the standard normal cumulative distribution function.

The optimal analog signal function, illustrated in figure 1, is

$$h_0 = h(g_i{}^{(1)}, +1, 0) = R(g_i{}^{(1)}, \delta) \qquad (10)$$

$$h_1 = h(g_i{}^{(1)}, +1, 1) = R(g_i{}^{(1)}, \epsilon) - 1$$

where, for some $A_4 > 0$ and $A_5 > 0$, $R(s,t) = A_4 \tanh(s) - A_5(s - g(t))$.

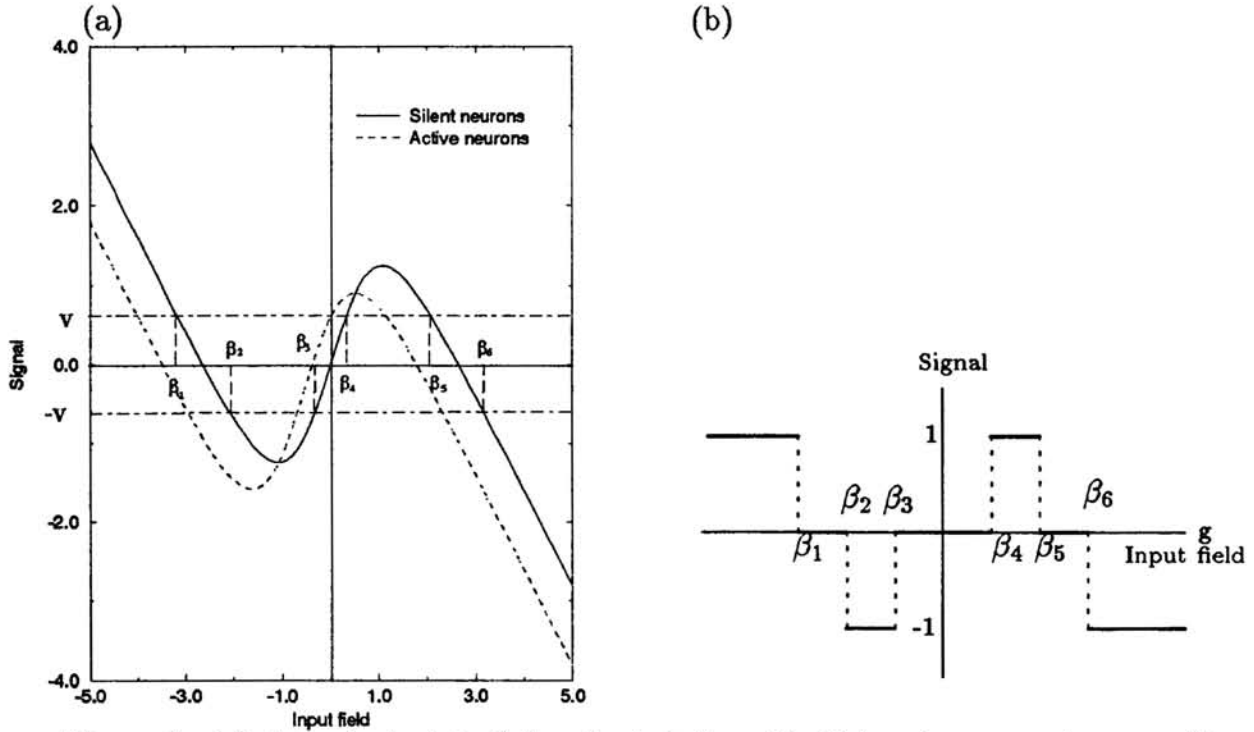

Figure 1: (a) A typical plot of the slanted sigmoid, Network parameters are $N = 5000$, $K = 3000$, $n_1 = 200$ and $m = 50$. (b) A sketch of its discretized version.

The nonmonotone form of these functions, illustrated in figure 1, is clear. Neurons that have already signalled +1 in the first iteration have a lesser tendency to send positive signals than quiescent neurons. The signalling of quiescent neurons which receive no prior information ($\delta = 0$) has a symmetric form. The optimal signal is shown to be essentially equal to the sigmoid modified by a correction term depending only on the current input field. In the limit of low memory load ($\epsilon/\sqrt{\alpha_1} \to \infty$), the best signal is simply a sigmoidal function of the generalized input field.

To obtain a discretized version of the slanted sigmoid, we let the signal be $sign(h(y))$ as long as $|h(y)|$ is big enough - where $h$ is the slanted sigmoid. The resulting signal, as a function of the generalized field, is (see figure 1a and 1b)

$$h_j(y) = \begin{cases} +1 & y < \beta_1^{(j)} \text{ or } \beta_4^{(j)} < y < \beta_5^{(j)} \\ -1 & y > \beta_6^{(j)} \text{ or } \beta_2^{(j)} < y < \beta_3^{(j)} \\ 0 & \text{otherwise} \end{cases} \qquad (11)$$

where $-\infty < \beta_1^{(0)} < \beta_2^{(0)} \le \beta_3^{(0)} < \beta_4^{(0)} \le \beta_5^{(0)} < \beta_6^{(0)} < \infty$ and $-\infty < \beta_1^{(1)} < \beta_2^{(1)} \le \beta_3^{(1)} < \beta_4^{(1)} \le \beta_5^{(1)} < \beta_6^{(1)} < \infty$ define, respectively, the firing pattern of the neurons that were silent or active in the first iteration. To find the best such discretized version of the optimal signal, we search numerically for the activity level $v$ which maximizes performance. Every activity level $v$, used as a threshold on $|h(y)|$, defines the (at most) twelve parameters $\beta_i^{(j)}$ (which are identified numerically via the Newton-Raphson method) as illustrated in figure 1b.

## 4    Numerical Results

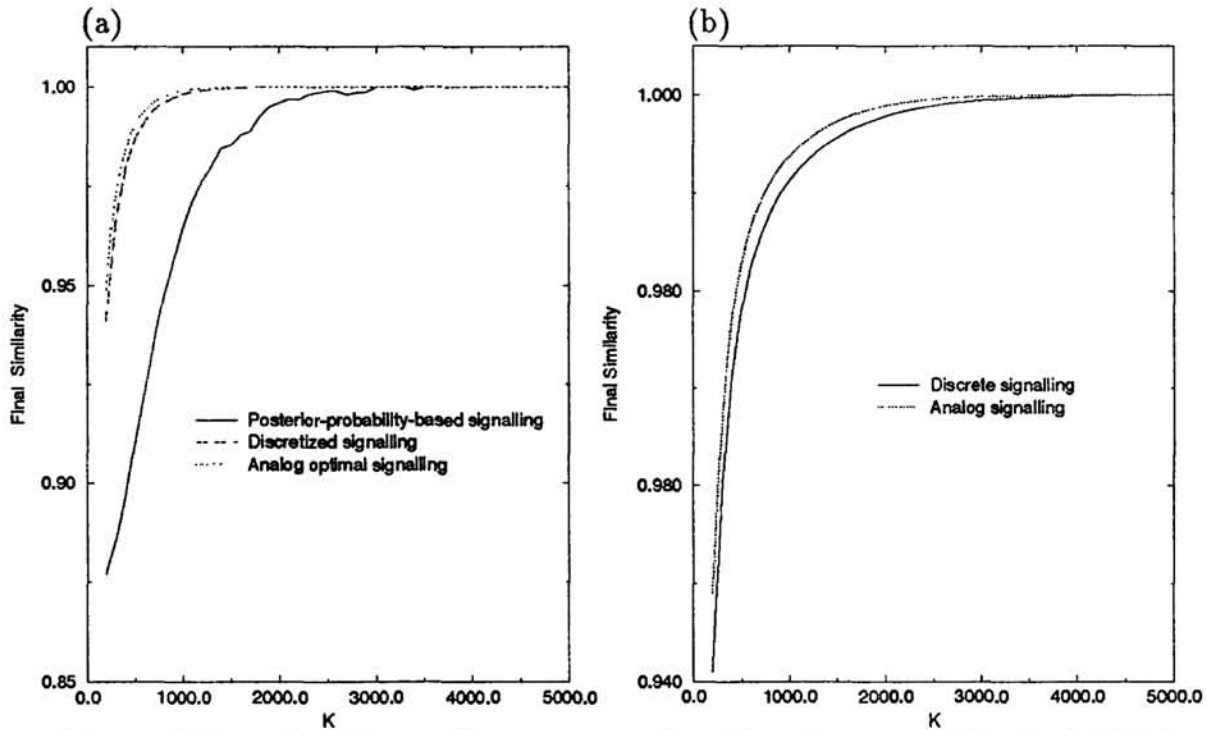

Figure 2: Two-iteration performance as a function of connectivity $K$. (a) Network parameters are $N = 5000$, $n_1 = 200$, and $m = 50$. All neurons receive their input state with similar initial overlap $\epsilon = \delta = 0.5$. (b) Network parameters are $N = 5000$, $m = 50$, $n_1 = 200$, $\epsilon = 0.5$ and $\delta = 0$.

Using the formulation presented in the previous section, we investigated numerically the two-iteration performance achieved in several network architectures with optimal analog signalling and its discretization. Already in small scale networks of a few hundred neurons our theoretical calculations correspond fairly accurately with

simulation results. First we repeat the example of a cortical-like network investigated in M & R, but now with optimal analog and discretized signalling. The nearly identical marked superiority of optimal analog and discretized dynamics over the previous, posterior-probability-based signalling is evident, as shown in figure 2 (a). While low activity is enforced in the first iteration, the number of neurons allowed to become active in the second iteration is not restricted, and best performance is typically achieved when about 70% of the neurons in the network are active (both with optimal signalling and with the previous, heuristic signalling).

Figure 2 (b) displays the performance achieved in the same network, when the input signal is applied only to the small fraction (4%) of neurons which are active in the first iteration (expressing possible limited resources of input information). We see that (for $K > 1000$) near perfect final similarity is achieved even when the 96% initially quiescent neurons get no initial clue as to their true memory state, if no restrictions are placed on the second iteration activity level.

Next we have fixed the value of $\omega = \frac{\epsilon}{\sqrt{\alpha_1}} = 1$, and contrasted the case ($n_1 = 200, \epsilon = 0.5$) of figure 2 (b) with ($n_1 = 50, \epsilon = 1$). The overall initial similarity under ($n_1 = 50, \epsilon = 1$) is only half its value under ($n_1 = 200, \epsilon = 0.5$). In spite of this, we have found that it achieves a slightly higher final similarity. This supports the idea that the input pattern should not be applied as the conventional uniformly distorted version of the correct memory, but rather as a less distorted pattern applied only to a small subset of the neurons.

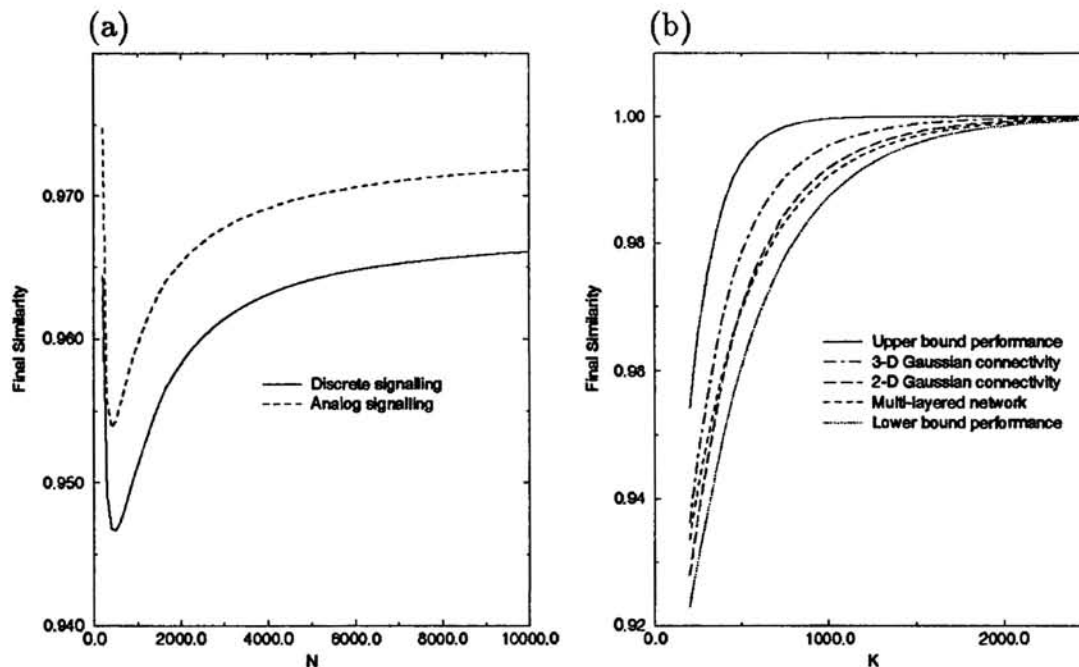

Figure 3: (a) Two-iteration performance in a full-activity network as a function of network size $N$. Network parameters are $n_1 = K = 200$, $m = 40$ and $\epsilon = 0.5$. (b) Two-iteration performance achieved with various network architectures, as a function of the network connectivity $K$. Network parameters are $N = 5000$, $n_1 = 200$, $m = 50$, $\epsilon = 0.5$ and $\delta = 0$.

Figure 3 (a) illustrates the performance when connectivity and the number of sig-

nals received by each neuron are held fixed, but the network size is increased. A region of decreased performance is evident at mid-connectivity ($K \approx N/2$) values, due to the increased residual variance. Hence, for neurons capable of forming $K$ connections on the average, the network should either be fully connected or have a size $N$ much larger than $K$. Since (unavoidable eventually) synaptic deletion would sharply worsen the performance of fully connected networks, cortical ANNs should indeed be sparsely connected. The final similarity achieved in the fully connected network (with $N = K = 200$) should be noted. In this case, the memory load (0.2) is significantly above the critical capacity of the Hopfield network, but optimal history-dependent dynamics still manage to achieve a rather high two-iterations similarity (0.975) from initial similarity 0.75. This is in agreement with the findings of [Morita, 1993, Yoshizawa *et al.*, 1993], who show that nonmonotone dynamics increase capacity.

Figure 3 (b) illustrates the performance achieved with various network architectures, all sharing the same network parameters $N, K, m$ and input similarity parameters $n_1, \epsilon, \delta$, but differing in the spatial organization of the neurons' synapses. As evident, even in low-activity sparse-connectivity conditions, the decrease in performance with Gaussian connectivity (in relation, say, to the upper bound) does not seem considerable. Hence, history-dependent ANNs can work well in a cortical-like architecture.

## 5  Summary

The main results of this work are as follows:

- The Bayesian framework gives rise to the slanted-sigmoid as the optimal signal function, displaying the non monotone shape proposed by [Morita, 1993]. It also offers an intuitive explanation of its form.

- Martingale arguments show that similarity under Bayesian dynamics persistently increases. This makes our two-iteration results a lower bound for the final similarity achievable in ANNs.

- The possibly asymmetric form of the function, where neurons that have been silent in the previous iteration have an increased tendency to fire in the next iteration versus previously active neurons, is reminiscent of the bi-threshold phenomenon observed in biological neurons [Tam, 1992].

- In the limit of low memory load the best signal is simply a sigmoidal function of the generalized input field.

- In an efficient associative network, input patterns should be applied with high fidelity on a small subset of neurons, rather than spreading a given level of initial similarity as a low fidelity stimulus applied to a large subset of neurons.

- If neurons have some restriction on the number of connections they may form, such that each neuron forms some $K$ connections on the average, then efficient ANNs, converging to high final similarity within few iterations, should be sparsely connected.

- With a properly tuned signal function, cortical-like Gaussian-connectivity ANNs perform nearly as well as randomly-connected ones.

- Investigating the $0, 1$ (silent, firing) formulation, there seems to be an interval such that only neurons whose field values are greater than some low threshold and smaller than some high threshold should fire. This seemingly bizarre behavior may correspond well to the behavior of biological neurons; neurons with very high field values have most probably fired constantly in the previous 'iteration', and due to the effect of neural adaptation are now silenced.

## Footnotes

*Currently in the Dept. of Computer science, University of Maryland

# References

[Braitenberg and Schuz, 1991] V. Braitenberg and A. Schuz. *Anatomy of the Cortex: Statistics and Geometry.* Springer-Verlag, 1991.

[Connors and Gutnick, 1990] B.W. Connors and M.J. Gutnick. Intrinsic firing patterns of diverse neocortical neurons. *TINS*, 13(3):99–104, 1990.

[Hopfield, 1982] J.J. Hopfield. Neural networks and physical systems with emergent collective abilities. *Proc. Nat. Acad. Sci. USA*, 79:2554, 1982.

[Lytton, 1991] W. Lytton. Simulations of cortical pyramidal neurons synchronized by inhibitory interneurons. *J. Neurophysiol.*, 66(3):1059–1079, 1991.

[Meilijson and Ruppin, 1993] I. Meilijson and E. Ruppin. History-dependent attractor neural networks. *Network*, 4:1–28, 1993.

[Morita, 1993] M. Morita. Associative memory with nonmonotone dynamics. *Neural Networks*, 6:115–126, 1993.

[Tam, 1992] David C. Tam. Signal processing in multi-threshold neurons. In T. McKenna, J. Davis, and S.F. Zornetzer, editors, *Single neuron computation*, pages 481–501. Academic Press, 1992.

[Yoshizawa *et al.*, 1993] S. Yoshizawa, M. Morita, and S.-I. Amari. Capacity of associative memory using a nonmonotonic neuron model. *Neural Networks*, 6:167–176, 1993.